# Information Capacity and Robustness of Stochastic Neuron Models

**Elad Schneidman**   **Idan Segev**   **Naftali Tishby**
Institute of Computer Science,
Department of Neurobiology and
Center for Neural Computation,
Hebrew University
Jerusalem 91904, Israel
*{elads,tishby}@cs.huji.ac.il, idan@lobster.ls.huji.ac.il*

## Abstract

The reliability and accuracy of spike trains have been shown to depend on the nature of the stimulus that the neuron encodes. Adding ion channel stochasticity to neuronal models results in a macroscopic behavior that replicates the input-dependent reliability and precision of real neurons. We calculate the amount of information that an ion channel based stochastic Hodgkin-Huxley (HH) neuron model can encode about a wide set of stimuli. We show that both the information rate and the information per spike of the stochastic model are similar to the values reported experimentally. Moreover, the amount of information that the neuron encodes is correlated with the amplitude of fluctuations in the input, and less so with the average firing rate of the neuron. We also show that for the HH ion channel density, the information capacity is robust to changes in the density of ion channels in the membrane, whereas changing the ratio between the $Na^+$ and $K^+$ ion channels has a considerable effect on the information that the neuron can encode. Finally, we suggest that neurons may maximize their information capacity by appropriately balancing the density of the different ion channels that underlie neuronal excitability.

## 1   Introduction

The capacity of neurons to encode information is directly connected to the nature of spike trains as a code. Namely, whether the fine temporal structure of the spike train carries information or whether the fine structure of the train is mainly noise (see e.g. [1, 2]). Experimental studies show that neurons *in vitro* [3, 4] and *in vivo* [5, 6, 7], respond to fluctuating inputs with repeatable and accurate spike trains, whereas slowly varying inputs result in lower repeatability and 'jitter' in the spike timing. Hence, it seems that the nature of the code utilized by the neuron depends on the input that it encodes [3, 6].

Recently, we suggested that the biophysical origin of this behavior is the stochas-

ticity of single ion channels. Replacing the average conductance dynamics in the Hodgkin-Huxley (HH) model [8], with a stochastic channel population dynamics [9, 10, 11], yields a stochastic neuron model which replicates rather well the spike trains' reliability and precision of real neurons [12]. The stochastic model also shows subthreshold oscillations, spontaneous and missing spikes, all observed experimentally. Direct measurement of membranal noise has also been replicated successfully by such stochastic models [13]. Neurons use many tens of thousands of ion channels to encode the synaptic current that reaches the soma into trains of spikes [14]. The number of ion channels that underlies the spike generation mechanism, and their types, depend on the activity of the neuron [15, 16]. It is yet unclear how such changes may affect the amount and nature of the information that neurons encode.

Here we ask what is the information encoding capacity of the stochastic HH model neuron and how does this capacity depend on the densities of different of ion channel types in the membrane. We show that both the information rate and the information per spike of the stochastic HH model are similar to the values reported experimentally and that neurons encode more information about highly fluctuating inputs. The information encoding capacity is rather robust to changes in the channel densities of the HH model. Interestingly, we show that there is an optimal channel population size, around the natural channel density of the HH model. The encoding capacity is rather sensitive to changes in the distribution of channel types, suggesting that changes in the population ratios and adaptation through channel inactivation may change the information content of neurons.

## 2 The Stochastic HH Model

The stochastic HH (SHH) model expands the classic HH model [8], by incorporating the stochastic nature of single ion channels [9, 17]. Specifically, the membrane voltage dynamics is given by the HH description, namely,

$$c_m \frac{dV}{dt} = -g_L(V - V_L) - g_K(V, t)(V - V_K) - g_{Na}(V, t)(V - V_{Na}) + I \qquad (1)$$

where $V$ is the membrane potential, $V_L$, $V_K$ and $V_{Na}$ are the reversal potentials of the leakage, potassium and sodium currents, respectively, $g_L$, $g_K(V, t)$ and $g_{Na}(V, t)$ are the corresponding ion conductances, $c_m$ is the membrane capacitance and $I$ is the injected current. The ion channel stochasticity is introduced by replacing the equations describing the ion channel conductances with explicit voltage-dependent Markovian kinetic models for single ion channels [9, 10]. Based on the activation and inactivation variables of the deterministic HH model, each $K^+$ channel can be in one of five different states, and the rates for transition between these states are given in the following diagram,

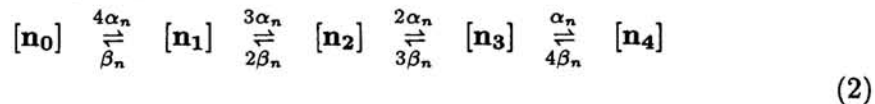

$$(2)$$

where $[\mathbf{n_j}]$ refers to the number of channels which are currently in the state $n_j$. Here $[\mathbf{n_4}]$ labels the single open state of a potassium channel, and $\alpha_n$, $\beta_n$, are the voltage-dependent rate-functions in the HH formalism. A similar model is used for the $Na^+$ channel (The $Na^+$ kinetic model has 8 states, with only one open state, see [12] for details).

The potassium and sodium membrane conductances are given by,

$$g_K(V, t) = \gamma_K [\mathbf{n_4}] \qquad g_{Na}(V, t) = \gamma_{Na} [\mathbf{m_3 h_1}] \qquad (3)$$

where $\gamma_K$ and $\gamma_{Na}$ are the conductances of an ion channel for the $K^+$ and $Na^+$ respectively. We take the conductance of a single channel to be $20\,pS$ [14] for both the

$K^+$ and $Na^+$ channel types [1]. Each of the ion channels will thus respond stochastically by closing or opening its 'gates' according to the kinetic model, fluctuating around the average expected behavior. Figure 1 demonstrates the effect of the ion

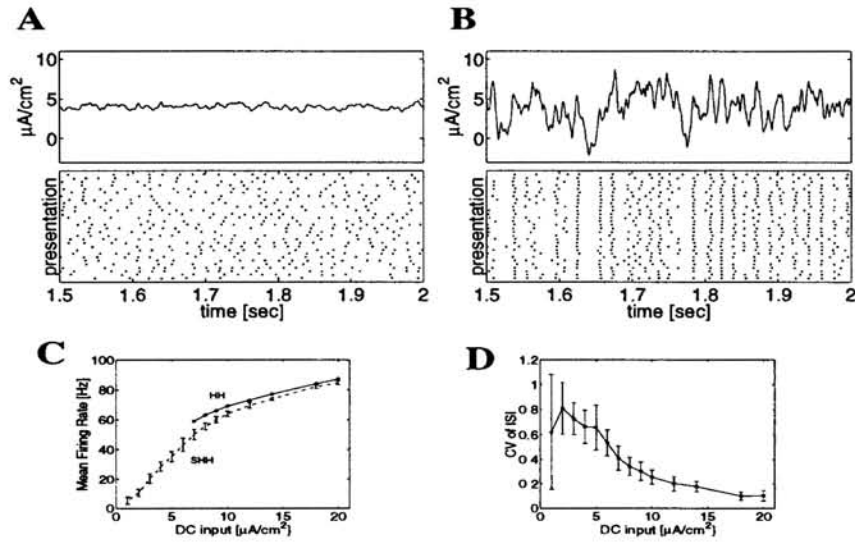

Figure 1: Reliability of firing patterns in a model of an isopotential Hodgkin-Huxley membrane patch in response to different current inputs. (A) Injecting a slowly changing current input (low-pass Gaussian white noise with a mean $\eta = 8\,\mu A/cm^2$, and standard deviation $\sigma = 1\,\mu A/cm^2$ which was convolved with an 'alpha-function' with a time constant $\tau_\alpha = 3\,msec$, top frame), results in high 'jitter' in the timing of the spikes (raster plots of spike responses, bottom frame). (B) The same patch was again stimulated repeatedly, with a highly fluctuating stimulus ($\eta = 8\,\mu A/cm^2$, $\sigma = 7\,\mu A/cm^2$ and $\tau_\alpha = 3\,msec$, top frame) The 'jitter' in spike timing is significantly smaller in B than in A (i.e. increased reliability for the fluctuating current input). Patch area used was $200\,\mu m^2$, with $3,600\,K^+$ channels and $12,000\,Na^+$ channels. (Compare to Fig.1 in see [3]). (C) Average firing rate in response to DC current input of both the HH and the stochastic HH model. (D) Coefficient of variation of the inter spike interval of the SHH model in response to DC inputs, giving values which are comparable to those observed in real neurons

channel stochasticity, showing the response of a $200\,\mu m^2$ SHH isopotential membrane patch (with the 'standard' SHH channel densities) to repeated presentation of suprathreshold current input. When the same slowly varying input is repeatedly presented (Fig. 1A), the spike trains are very different from each other, i.e., spike firing time is unreliable. On the other hand, when the input is highly fluctuating (Fig. 1B), the reliability of the spike timing is relatively high. The stochastic model thus replicates the input-dependent reliability and precision of spike trains observed in pyramidal cortical neurons [3]. As for cortical neurons, the *Repeatability* and *Precision* of the spike trains of the stochastic model (defined in [3]) are strongly correlated with the fluctuations in the current input and may get to sub-millisecond precision [12]. The f-I curve of the stochastic model (Fig. 1C) and the coefficient of variation (CV) of the inter-spike intervals (ISI) distribution for DC inputs (Fig. 1D) are both similar to the behavior of cortical neurons *in vivo* [18], in clear contrast to the deterministic model [2]

## 3   The Information Capacity of the SHH Neuron

Expanding the *Repeatability* and *Precision* measures [3], we turn to quantify how much information the neuron model encodes about the stimuli it receives. We thus present the model with a set of 'representative' input current traces, and the amount of information that the respective spike trains encode is calculated.

Following Mainen and Sejnowski [3], we use a set of input current traces which imitate the synaptic current that reaches the soma from the dendritic tree. We convolve a Gaussian white noise trace (with a mean current $\eta$ and standard deviation $\sigma$) with an alpha function (with a $\tau_\alpha = 3\ msec$). Six different mean current values are used ($\eta = 0, 2, 4, 6, 8, 10\ \mu A/cm^2$) , and five different std values ($\sigma = 1, 3, 5, 7, 9\ \mu A/cm^2$), yielding a set of 30 input current traces (each is 10 seconds long). This set of inputs is representative of the wide variety of current traces that neurons might encounter under *in vivo* conditions in the sense that the average firing rates for this set of inputs which range between $2 - 70\ Hz$ (not shown).

We present these input traces to the model, and calculate the amount of information that the resulting spike trains convey about each input, following [6, 19]. Each input is presented repeatedly and the resulting spike trains are discretized in $\Delta \tau$ bins, using a sliding 'window' of size $T$ along the discretized sequence. Each train of spikes is thus transformed into a sequence of K-letter 'words' ($K = T/\Delta\tau$), consisting of 0's (no spike) and 1's (spike). We estimate $P(W)$, the probability of the word $W$ to appear in the spike trains, and then compute the entropy rate of its total word distribution,

$$H_{total} = - \sum_{W} P(W) \log_2 P(W) \quad bits/word \tag{4}$$

which measures the capacity of information that the neuron spike trains hold [20, 6, 19]. We then examine the set of words that the neuron model used at a particular time $t$ over all the repeated presentations of the stimulus, and estimate $P(W|t)$, the time-dependent word probability distribution. At each time $t$ we calculate the time-dependent entropy rate, and then take the average of these entropies

$$H_{noise} = \langle - \sum_{W} P(W|t) \log_2 P(W|t) \rangle_t \quad bits/word \tag{5}$$

where $\langle\ldots\rangle_t$ denotes the average over all times $t$. $H_{noise}$ is the noise entropy rate, which measures how much of the fine structure of the spike trains of the neuron is just noise. After performing the calculation for each of the inputs, using different word sizes [3], we estimate the limit of the total entropy and noise entropy rates at $T \to \infty$, where the entropies converge to their real values (see [19] for details) .

Figure 2A shows the total entropy rate of the responses to the set of stimuli, ranging from 10 to 170 *bits/sec*. The total entropy rate is correlated with the firing rates of the neuron (not shown). The noise entropy rate however, depends in a different way on the input parameters: Figure 2B shows the noise entropy rate of the responses to the set of stimuli, which may get up to 100 *bits/sec*. Specifically, for inputs with high mean current values and low fluctuation amplitude, many of the spikes are

---

of ion channels which are open near the spike firing threshold is rather small [12]. The fluctuations in this small number of open channels near firing threshold give rise to the input-dependent reliability of the spike timing.

[3]the bin size $\tau = 2\ msec$ has been set to be small enough to keep the fine temporal structure of the spike train within the word sizes used, yet large enough to avoid undersampling problems

just noise, even if the mean firing rate is high. The difference between the neuron's entropy rate (the total capacity of information of the neuron's spike train) and the noise entropy rate, is exactly the average rate of information that the neuron's spike trains encode about the input, $I(stimulus, spike\ train) = H_{total} - H_{noise}$ [20, 6], this is shown in Figure 2C. The information rate is more sensitive to the size of

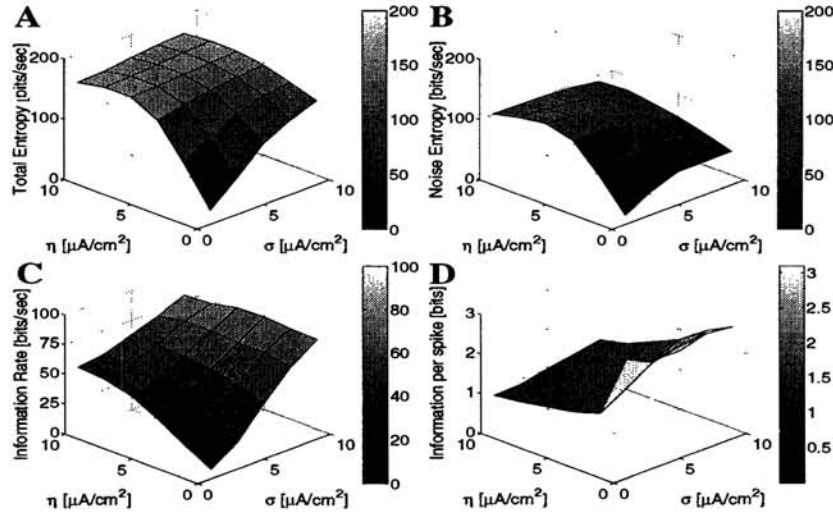

Figure 2: Information capacity of the SHH model. (A) The total spike train entropy rate of the SHH model as a function of $\eta$, the current input mean, and $\sigma$, the standard deviation (see text for details). Error bar values of this surface as well as for the other frames range between $1 - 6\%$ (not shown). (B) Noise entropy rate as a function of the current input parameters. (C) The information rate about the stimulus in the spike trains, as a function of the input parameters, calculated by subtracting noise entropy from the total entropy (note the change in grayscale in C and D). (D) Information per spike as a function of the input parameters, which is calculated by normalizing the results shown in C by the average firing rate of the responses to each of the inputs.

fluctuations in the input than to the mean value of the current trace (as expected, from the reliability and precision of spike timing observed *in vitro* [3] and *in vivo* [6] as well as in simulations [12]). The dependence of the neural code on the input parameters is better reflected when calculating the average amount of information per spike that the model gives for each of the inputs (Fig. 2D) (see for comparison the values for the Fly's H1 neuron [6]).

## 4    The effect of Changing the Neuron Parameters on the Information Capacity

Increasing the density of ion channels in the membrane compared to the 'standard' SHH densities, while keeping the ratio between the $K^+$ and $Na^+$ channels fixed, only diminishes the amount of information that the neuron encodes about any of the inputs in the set. However, the change is rather small: Doubling the channel density decreases the amount of information by $5 - 25\%$ (Fig. 3A), depending on the specific input. Decreasing the channel densities of both types, results in encoding more information about certain stimuli and less about others. Figure 3B shows that having half the channel densities would result with in 10% changes in the information in both directions. Thus, the information rates conveyed by the stochastic model are robust to changes in the ion channel density. Similar robustness (not shown) has been observed for changes in the membrane area (keeping channel

density fixed) and in the temperature (which effects the channel kinetics). However,

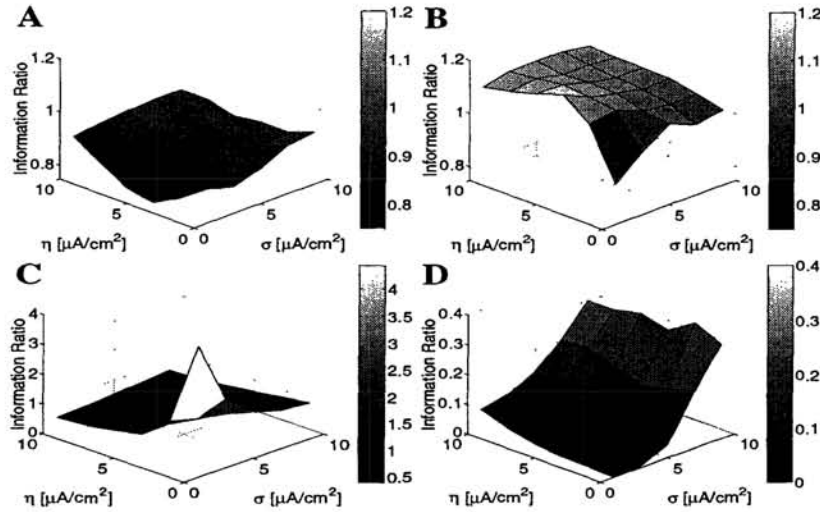

Figure 3: The effect of changing the ion channel densities on the information capacity. (A) The ratio of the information rate of the SHH model with twice the density of the 'standard' SHH densities divided by the information rate of the mode with 'standard' SHH densities. (B) As in A, only for the SHH model with half the 'standard' densities. (C) The ratio of the info rate of the SHH model with twice as many $Na^+$ channels, divided by the info rate of the standard SHH $Na^+$ channel density, where the $K^+$ channel density remains untouched (note the change in graycale in C and D). (D) As in C, only for the SHH model with the number of $Na^+$ channels reduced by half.

changing the density of the $Na^+$ channels alone has a larger impact on the amount of information that the neuron conveys about the stimuli. Increasing $Na^+$ channel density by a factor of two results in less information about most of the stimuli, and a gain in a few others (Fig. 3C). However, reducing the number of $Na^+$ channels by half results in drastic loss of information for all of the inputs (Fig. 3D).

# 5    Discussion

We have shown that the amount of information that the stochastic HH model encodes about its current input is highly correlated with the amplitude of fluctuations in the input and less so with the mean value of the input. The stochastic HH model, which incorporates ion channel noise, closely replicates the input-dependent reliability and precision of spike trains observed in cortical neurons. The information rates and information per spike are also similar to those of real neurons. As in other biological systems (e.g., [21]), we demonstrate robustness of macroscopic performance to changes in the cellular properties – the information coding rates of the SHH model are robust to changes in the ion channels densities as well as in the area of the excitable membrane patch and in the temperature (kinetics) of the channel dynamics. However, the information coding rates are rather sensitive to changes in the ratio between the densities of different ion channel types, suggests that the ratio between the density of the $K^+$ channels and the $Na^+$ channels in the 'standard' SHH model may be optimal in terms of the information capacity. This may have important implications on the nature of the neural code under adaptation and learning. We suggest that these notions of optimality and robustness may be a key biophysical principle of the operation of real neurons. Further investigations should take into account the activity-dependent nature of the channels and the

neuron [15, 16] and the notion of local learning rules which could modify neuronal and suggest local learning rules as in [22].

## Acknowledgements

This research was supported by a grant from the Ministry of Science, Israel.

## Footnotes

[1]The number of channels is thus the ratio between the total conductance of a single type of ion channels and the single channel conductance, and so the 'standard' SHH densities will be 60 $Na^+$ and 18 $Na^+$ channels per $\mu m^2$.

[2]Although the total number of channels in the model is very large, the microscopic level ion channel noise has a macroscopic effect on the spike train reliability, since the number

## References

[1] Rieke F., Warland D., de Ruyter van Steveninck R., and Bialek W. *Spike: Exploring the Neural Code.* MIT Press, 1997.

[2] Shadlen M. and Newsome W. Noise, neural codes and cortical organization. *Curr. Opin. Neurobiol.*, 4:569–579, 1994.

[3] Mainen Z. and Sejnowski T. Reliability of spike timing in neocortical neurons. *Science*, 268:1503–1508, 1995.

[4] Nowak L., Sanches-Vives M., and McCormick D. Influence of low and high frequency inputs on spike timing in visual cortical neurons. *Cerebral Cortex*, 7:487–501, 1997.

[5] Bair W. and Koch C. Temporal precision of spike trains in extrastriate cortex of the behaving macaque monkey. *Neural Comp.*, 8:1185–1202, 1996.

[6] de Ruyter van Steveninck R., Lewen G., Strong S., Koberle R., and Bialek W. Reproducibility and variability in neural spike trains. *Science*, 275:1805–1808, 1997.

[7] Reich D., Victor J., Knight B., Ozaki T., and Kaplan E. Response variability and timing precision of neuronal spike trains in vivo. *J. Neurophysiol.*, 77:2836:2841, 1997.

[8] Hodgkin A. and Huxley A. A quantitative description of membrane current and its application to conduction and excitation in nerve. *J. Physiol.*, 117:500–544, 1952.

[9] Fitzhugh R. A kinetic model of the conductance changes in nerve membrane. *J. Cell. Comp. Physiol.*, 66:111–118, 1965.

[10] DeFelice L. *Introduction to Membrane Noise.* Perseus Books, 1981.

[11] Skaugen E. and Walløe L. Firing behavior in a stochastic nerve membrane model based upon the Hodgkin-Huxley equations. *Acta Physiol. Scand.*, 107:343–363, 1979.

[12] Schneidman E., Freedman B., and Segev I. Ion channel stochasticity may be critical in determining the reliability and precision of spike timing. *Neural Comp.*, 10:1679–1704, 1998.

[13] White J., Klink R., Alonso A., and Kay A. Noise from voltage-gated channels may influence neuronal dynamics in the entorhinal cortex. *J Neurophysiol*, 80:262-9, 1998.

[14] Hille B. *Ionic Channels of Excitable Membrane.* Sinauer Associates, 2nd ed., 1992.

[15] Marder E., Abbott L., Turrigiano G., Liu Z., and Golowasch J. Memory from the dynamics of intrinsic membrane currents. *Proc. Natl. Acad. Sci.*, 93:13481-6, 1996.

[16] Toib A., Lyakhov V., and Marom S. Interaction between duration of activity and rate of recovery from slow inactivation in mammalian brain $Na^+$ channels. *J Neurosci.*, 18:1893–1903, 1998.

[17] Strassberg A. and DeFelice L. Limits of the HH formalism: Effects of single channel kinetics on transmembrane voltage dynamics. *Neural Comp.*, 5:843–856, 1993.

[18] Softky W. and Koch C. The highly irregular firing of cortical cells is inconsistent with temporal integration of random EPSPs. *J. Neurosci.*, 13:334–350, 1993.

[19] Strong S., Koberle R., de Ruyter van Steveninck R., and Bialek W. Entropy and information in neural spike trains. *Phys. Rev. Lett.*, 80:197–200, 1998.

[20] Cover T.M. and Thomas J.A. *Elements of Information Theory.* Wiley, 1991.

[21] Barkai N. and Leibler S. Robustness in simple biochemical networks. *Nature*, 387:913–917, 1997.

[22] Stemmler M. and Koch C. How voltage-dependent conductances can adapt to maximize the information encoded by neuronal firing rate. *Nat. Neurosci.*, 2:521-7, 1999.